# Predicting Brain States from fMRI Data: Incremental Functional Principal Component Regression

**S. Ghebreab**
ISLA/HCS lab, Informatics Institute
University of Amsterdam, The Netherlands
ghebreab@science.uva.nl

**A.W.M. Smeulders**
ISLA lab, Informatics Institute
University of Amsterdam, The Netherlands
smeulders@science.uva.nl

**P. Adriaans**
HCS lab, Informatics Institute
University of Amsterdam, The Netherlands
pietera@science.uva.nl

## Abstract

We propose a method for reconstruction of human brain states directly from functional neuroimaging data. The method extends the traditional multivariate regression analysis of discretized fMRI data to the domain of stochastic functional measurements, facilitating evaluation of brain responses to complex stimuli and boosting the power of functional imaging. The method searches for sets of voxel time courses that optimize a multivariate functional linear model in terms of $R^2$-statistic. Population based incremental learning is used to identify spatially distributed brain responses to complex stimuli without attempting to localize function first. Variation in hemodynamic lag across brain areas and among subjects is taken into account by voxel-wise non-linear registration of stimulus pattern to fMRI data. Application of the method on an international test benchmark for prediction of naturalistic stimuli from new and unknown fMRI data shows that the method successfully uncovers spatially distributed parts of the brain that are highly predictive of a given stimulus.

## 1 Introduction

To arrive at a better understanding of human brain function, functional neuroimaging traditionally studies the brain's responses to controlled stimuli. Controlled stimuli have the benefit of leading to clear and often localized response signals in fMRI as they are specifically designed to affect only certain brain functions. The drawback of controlled stimuli is that they are a reduction of reality: one cannot be certain whether the response is due to the reduction or due to the stimulus. Naturalistic stimuli open the possibility to avoid the question whether the response is due to the reduction or the signal. Naturalistic stimuli, however, carry a high information content in their spatio-temporal structure that is likely to instigate complex brain states. The immediate consequence hereof is that one faces the task of isolating relevant responses amids complex patterns.

To reveal brain responses to naturalistic stimuli, advanced signal processing methods are required that go beyond conventional mass univariate data analysis. Univariate techniques generally lack sufficient power to capture the spatially distributed response of the brain to naturalistic stimuli. Multivariate pattern techniques, on the other hand, have the capacity to identify patterns of information when they are present across the full spatial extent of the brain without attempting to localize func-

tion. Here, we propose a multivariate pattern analysis approach for predicting naturalistic stimuli on the basis of fMRI data. Inverting the task from correlating stimuli with fMRI data to predicting stimuli from fMRI data makes it easier to evaluate brain responses to naturalistic stimuli and may extend the power of functional imaging substantially [1].

Various multivariate approaches for reconstruction of brain states directly from fMRI measurements have recently been proposed. In most of these approaches, a classifier is trained directly on the fMRI data to discriminate between known different brain states. This classifier is then used to predict brain states on the basis of new and unknown fMRI data alone. Such approaches have been used to predict what percept is dominant in a binocular rivalry protocol [2], what the orientation is of structures subjects are viewing [3] and what the semantic category is of objects [4] and words [5] subjects see on a screen. In one competition [6], participants trained pattern analyzers on fMRI of subjects viewing two short movies as well as on the subject's movie feature ratings. Then participants employed the analyzers to predict the experience of subjects watching a third movie based purely on fMRI data. Very accurate predictions were reported for identifying the presence of specific time varying movie features (e.g. faces, motion) and the observers who coded the movies [7].

We propose an incremental multivariate linear modeling approach for functional covariates, i.e. where both the fMRI data and external stimuli are continuous. This approach differs fundamentally from existing multivariate linear approaches (e.g. [8]) that instantly fit a given model to the data within the linear framework under the assumption that both the data and the model are discrete. Contemporary neuroimaging studies increasingly use high-resolution fMRI to accurately capture continuous brain processes, frequently instigated by continuous stimulations. Hence, we propose the use of functional data analysis [9], which treats data, or the processes giving rise to them, as functions. This not only allows to overcome limitations in neuroimaing studies due to the large number of data points compared to the number of samples, but also allows to exploit the fact that functions defined on a specific domain form an inner product vector space, and in most circumstances can be treated algebraically like vectors [10].

We extend classical multivariate regression analysis of fMRI data [11] to stochastic functional measurements. We show that, cast into an incremental pattern searching framework, functional multivariate regression provides a powerful technique for fMRI-based prediction of naturalistic stimuli.

## 2  Method

In the remainder, we consider stimuli data and data produced by fMRI scanners as continuous functions of time, sampled at the scan interval and subject to observational noise. We treat the data within a functional linear model where both the predictant and predictor are functional, but where the design matrix that takes care of the linear mapping between the two is vectorial.

### 2.1  The Predictor

The predictor data are derived directly from the four-dimensional fMRI data $I(\mathbf{x}, t)$, where $\mathbf{x} \in \mathfrak{R}^3$ denotes the spatial position of a voxel and $t$ denotes its temporal position. We represent each of the $S$ voxel time courses in functional form by $f_s(t)$, with $t$ denoting the continuous path parameter and $s = 1, ..., S$. Rather than directly using voxel time courses for prediction, we use their principal components to eliminate collinearity in the predictor set. Following [10], we use functional principal component analysis. Viviani et al. [10] showed that functional principal components analysis is more effective than is its ordinary counterpart in recovering the signal of interest in fMRI data, even if limited or no prior knowledge of the hemodynamic function or experimental design is specified. In contrast to [10], however, our approach incrementally zooms in on stimuli-related voxel time courses for dimension reduction (see section 2.5).

Given the set of $S$ voxel time courses represented by the vector of functionals $\mathbf{f}(t) = [f_1(t), ..., f_S(t)]^T$, functional principal components analysis extracts main modes of variation in $\mathbf{f}(t)$. The number of modes to retain is determined from the proportion of the variance that needs to be explained. Assuming this is $Q$, the central concept is that of taking the linear combination

$$f_{sq} = \int_t f_s(t)\alpha_q(t)dt \tag{1}$$

where $f_{sq}$ is the principal component score value of voxel time course $f_s(t)$ in dimension $q$. Principal components $\alpha_q(t), q = 1, .., Q$ are sought for one-by-one by optimizing

$$\alpha_q(t) = \max_{\alpha_q^*(t)} \frac{1}{S} \sum_{s=1}^{S} f_{sq}^2 \qquad (2)$$

where $\alpha_q(t)$ is subject to the following orthonormal constraints

$$\int_t \alpha_q(t)^2 dt = 1 \qquad \int_t \alpha_k(t)\alpha_q(t)dt = 0, k \leq q. \qquad (3)$$

The mapping of $f_s(t)$ onto the subspace spanned by the first $Q$ principal component curves results in the vector of scalars $\mathbf{f}_s = [f_{s1}, ..., f_{sQ}]$. We define the $S \times Q$ matrix $\mathbf{F} = [\mathbf{f}_1, ..., \mathbf{f}_S]^T$ of principal components scores as our predictor data in linear regression. That is, we perform principal component regression with $\mathbf{F}$ as model, allowing to naturally deal with temporal correlations, multicollinearity and systematic signal variation.

## 2.2 The Predictand

We represent the stimulus pattern by the functional $g(t)$, $t$ being the continuous time parameter. We register $g(t)$ to each voxel time course $f_s(t)$ in order to be able to compare equivalent time points on stimulus and brain activity data. Alignment reduces to finding the warping function $\omega_s(t)$ that produces the warped stimulus function

$$g_s(t) = g(\omega_s(t)). \qquad (4)$$

The time warping function $\omega_s(t)$ is strictly monotonic, differentiable up to a certain order and takes care of a small shift and nonlinear transformation. A global alignment criteria and least squares estimation is used:

$$\omega_s(t) = \min_{\omega_s^*} \int_t (g(\omega_s^*(t)) - f_s(t))^2 dt. \qquad (5)$$

Registration of $g(t)$ to all voxel time courses $S$ results in predictand data $\mathbf{g}(t) = [g_1(t), ..., g_S(t)]^T$, where $g(t)$ is $g(t)$ registered onto voxel times-course $f(t)$. Our motivation for using voxel-wise registration over standard convolution of stimulus $g(t)$ with the hemodynamic reponse function, is the large variability in hemodynamic delays across brain regions and subjects. A non-linear warp of $g(t)$ does not guarantee an outcome that is associated with brain physiology, however it allows to capture unknown subtle localized variations in hemodynamic delays across brain regions and subjects.

## 2.3 The Model

We employ the predictor data to explain the predictand data within a linear modeling approach, i.e. our multivariate linear model is defined as

$$\mathbf{g}(t) = \mathbf{F}\boldsymbol{\beta}(t) + \boldsymbol{\epsilon}(t) \qquad (6)$$

with $\boldsymbol{\beta}(t) = [\beta_1(t), ..., \beta_Q(t)]^T$ being the $Q \times 1$ vector of regression functions. The regression functions are estimated by least squares minimization such that

$$\hat{\boldsymbol{\beta}}(t) = \min_{\boldsymbol{\beta}^*(t)} \int_t (\mathbf{g}(t) - \mathbf{F}\boldsymbol{\beta}^*(t))^2 dt, \qquad (7)$$

under the assumption that the residual functions $\boldsymbol{\epsilon}(t) = [\epsilon_1(t), ...., \epsilon_S(t)]^T$ are independent and normally distributed with zero mean. The estimated regression functions provide the best estimate of $\mathbf{g}(t)$ in least squares sense:

$$\hat{\mathbf{g}}(t) = \mathbf{F}\hat{\boldsymbol{\beta}}(t). \qquad (8)$$

Given a new (sub)set of voxel time courses, prediction of a stimulus pattern now reduces to computing the matrix of principal component scores from this new set and weighting these scores by the estimated regression functions $\hat{\boldsymbol{\beta}}(t)$.

## 2.4   The Objective

The overall fit of the model to the data is expressed in terms of adjusted $R^2$ statistic. The functional counterpart of the traditional $R^2$ is computed on the basis of $\mathbf{g}(t)$, its mean $\bar{g}(t)$ and its estimation $\hat{\mathbf{g}}(t)$. For the voxel set $S$,

$$\dot{g}_S(t) = \sum_{s=1}^{S} (g_s(t) - \bar{g}(t))^2 \tag{9}$$

$$\ddot{g}_S(t) = \sum_{s=1}^{S} (g_s(t) - \hat{g}_s(t))^2 \tag{10}$$

are derived, where the first term is the variation of the response about its mean and the second the error sum of squares function. The adjusted R-square function is then defined as

$$\mathcal{R}_S(t) = 1 - \frac{\ddot{g}_S(t)/S - Q - 1}{\dot{g}_S(t)/S - 1} \tag{11}$$

where degrees of freedom $S - Q - 1$ and $S - 1$ adjust the R-square. Our objective is to find the set of voxel time courses $\mathcal{S}$ defined as

$$\mathcal{S} = \max_{S^* \subset S} \int_t \mathcal{R}_{S^*}(t) dt \tag{12}$$

where $S^*$ denotes a subset of the entire collection of voxels time courses $S$ extracted from a single fMRI scan. That is, we aim at finding spatially distributed voxel responses $\mathcal{S}$ that best explain the naturalistic stimuli, without making any prior assumptions about location and size of voxel subsets.

## 2.5   The Search

In order to efficiently find the subset of voxels that maximizes Equation (12), we use Population-Based Incremental Learning (PBIL) [12], which combines Genetic Algorithms with Competitive Learning. The PBIL algorithm uses a probability vector to explore the space of solutions. It incrementally generates solutions by sampling from that probability vector, evaluates these solutions and selects promising ones to update the probability vector. Here, at increment $i$, the probability vector $\mathbf{p}^i = [p_1^i, ..., p_S^i]$ is used to generate a population of $N$ solutions $\mathbf{M}^i = [\mathbf{m}_1^i, ..., \mathbf{m}_N^i]$, where each member is an S-vector of binary values: $\mathbf{m}_n^i = [m_{n1}^i, ..., m_{nS}^i]$. A value of 1 for $m_{ns}$ means that for solution $n$ the corresponding voxel time course $f_s(t)$ is included in the predictor set, while a value 0 indicates exclusion. Each member $\mathbf{m}_n^i$ is evaluated in terms of its adjusted $R^2$ value, and the members with highest values form the joint probability vector $\mathbf{p}^*$. A new probability vector is subsequently constructed for the next generation via competitive learning:

$$\mathbf{p}^{i+1} = \gamma \mathbf{p}^i + (1 - \gamma)\mathbf{p}^*. \tag{13}$$

The learning parameter $\gamma$ controls the search: a low value enables to focus entirely on the most recent voxel subset while a low value ensures that previously selected voxel subsets are exploited. In order to ensure spatial coherence and limit computation load, we employ the PBIl algorithm not on single time courses, but on averages of spatial clusters of voxel time courses. That is, we first spatially cluster voxel locations as shown in Figure 1, then compute average time course for each cluster and then explore the averages via PBIL for model building.

## 2.6   The Prediction

The subset of voxel time courses that results from population based incremental learning defines the most predictive voxel locations and associated regression functions. Given new and spatially normalized fMRI data, represented by $\tilde{\mathbf{f}}(t) = [\tilde{f}_1(t), ..., \tilde{f}_S(t)]^T$, prediction of a stimulus then reduces to computing

$$\tilde{\mathbf{g}}(t) = \tilde{\mathbf{F}}\hat{\beta}(t). \tag{14}$$

In here, $\tilde{\mathbf{g}}(t)$ is the vector of predicted stimuli of which the mean is considered to be the sought stimulus. The matrix $\tilde{\mathbf{F}}$ is the principal component scores matrix obtained from performing functional principal components analysis on subset $\tilde{\mathbf{f}}_S(t)$, with $\mathcal{S}$ referring to the set of most predictive voxels as determined by training.

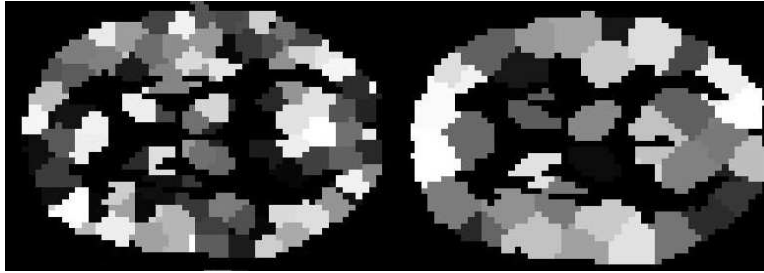

Figure 1: Examples of K-means clustering of voxel locations using Euclidean distance. Left: 1024-means clustering output. Right: 512-means clustering output. Different gray values indicate different clusters in a spatially normalized brain atlas.

## 3   Experiments and Results

### 3.1   Experiment

Evaluation of our method is done on a data subset from the 2006 Pittsburgh brain activity interpretation competition (PBAIC) [6, 7], involving fMRI scans of three different subjects and two movie sessions. In each session, a subject viewed a new Home Improvement sitcom movie for approximately 20 minutes. The 20-minute movie contained 5 interruptions where no video was present, only a white fixation cross on a black background. All three subjects watched the same two movies. The scans produced volumes with approximately 35,000 brain voxels, each approximately 3.28mm by 3.28mm by 3.5mm, with one volume produced every 1.75 seconds. These scans were preprocessed (motion correction, slice time correction, linear trend removal) and spatially normalized (non-linear registration to the Montreal Neurological Institute brain atlas).

After fMRI scanning, the three subjects watched the movie again to rate 30 movie features at time intervals corresponding to the fMRI scan rate. In our experiments, we focus on the 13 core movie features: *amusement, attention, arousal, body parts, environmental sounds, faces, food, language, laughter, motion, music, sadness and tools*. The real-valued ratings were convolved with a hemodynamic response function (HRF) modeled by two gamma functions, then subjected to voxel-wise non-linear registration as described in 2.2.

For training and testing our model, we removed parts corresponding with video presentations of a white fixation cross on a black background. Taking into account the hemodynamic lag, we divided each fMRI scan and each subject rating into 6 parts corresponding with the movie on parts. On average each movie part contained 105 discrete measurements. We then functionalized these parts by fitting a 30 coefficient B-spline to each voxel's discrete time course. This resulted in 18 data sets for training (3 subjects × 6 movie parts) and another 18 for testing. We used movie 1 data for training and movie 2 data for prediction, and vice versa. We performed data analysis at two levels. For each feature, first the individual brain scans were analyzed with our method, resulting in a first sifting of voxels. First-level analysis results for a given feature were then subjected to second level analysis to identify across subject predictive voxels. Pearson product-moment correlation coefficient between manual feature rating functions and the automatically predicted feature functions was used as an evaluation measure.

### 3.2   Results

All results were obtained with $Q = 4$ principal component dimensions, learning parameter value $\gamma = 0.6$ and K-means clustering with 1024 clusters for all movie features. These values for $Q$ and $\gamma$ produced overall highest average cross correlation value in a small parameter optimization experiment (data not shown here). Little performance differences were seen for various numbers of dimensions, indicating that the essential information can be captured with as little as 4 dimension. Significant performance differences across features, however, were observed for different learning parameter values, indicating considerable variation in brain response to distinct stimuli.

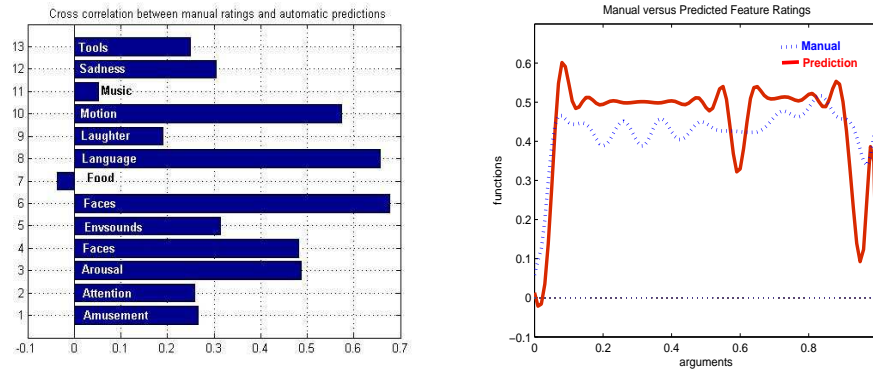

Figure 2: Left: normalized cross correlation values from cross-validation for 13 core movie features. Right: functionalized subject3 (solid red) and predicted (dotted blue) rating for the *language* feature of part 5 of movie 1.

Figure 2 (left) shows the average of $2 \times 18$ cross correlation coefficients from cross validation for all 13 movie features. For features *faces*, *language* and *motion* cross correlation values above 0.5 were obtained, meaning that there is a significant degree of match between the subject ratings and the predicted ratings. Reasonable predictions were also obtained for features *arousal* and *body parts*. Our results are consistent with top 3 rank entries of 2006 PBAIC in that features *faces* and *language* are reliably predicted. These entries used recurrent neural networks, ridge regression and a dynamic Gaussian Markov Random Field modeling on the entire test data benchmark, yielding across feature average cross correlations of: 0.49, 0.49 and 0.47 respectively. Here, the feature average cross correlation value based on the reduced training data set is 0.36. Note, that in the 2006 competition our method ranked first in the actor category [6]. We were able to accurately predict which actor the subjects were seeing purely based on fMRI scans [7].

The best single result, with highest cross correlation value of 0.76, was obtained for feature *language* of subject 3 watching part 5 of movie 1. For this feature, first level analysis of each of the 18 training data sets associated with movie 2 produced a total number of 1738 predictive voxels. In the second level analysis, these voxels were analyzed again to arrive at a reduced data set of 680 voxels for building the multivariate functional linear model and determining regression functions $\beta(t)$. For prediction of feature *language*, corresponding voxel time courses were extracted from the fMRI data of subject 3 watching movie 1 part 5, and weighted by $\beta(t)$. The manual rating of feature *language* of movie 1 part 5 by subject 3 and the average of the automatically predicted feature functions are shown in Figure 2 (right).

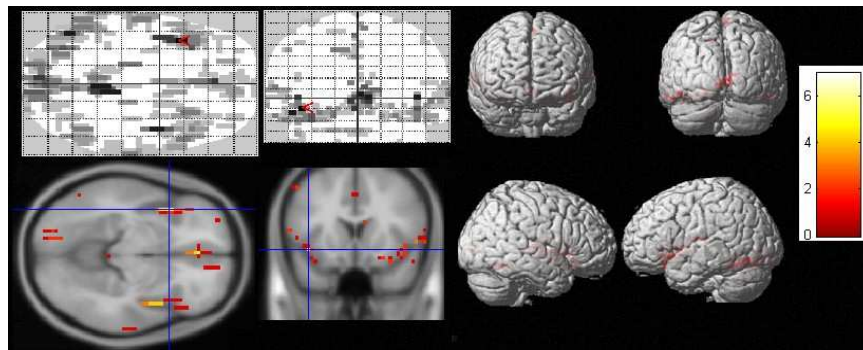

Figure 3: Glass view, gray level image with color overlay and surface rendering of 1738 voxels from first level analysis. Color denotes predictive power and cross hair shows most predictive location.

Figure 3 shows glass view, gray level image with color overlay and surface rendering of the 1738 voxels (approximately 40 clusters) from first level analysis. The cross hair shows the voxel location in Brodman area 47 that was found to be predictive across most subjects and movie parts: it was selected in 6 out of 18 training items (see color bar). The predictive locations correspond with the left and right inferior frontal gyrus, which are known to be involved in language processing. The distributed nature of these clusters is consistent with earlier findings that processing involved in language occurs in diffuse brain regions, including primary auditory and visual cortex, frontal regions in the left and right hemisphere, in homologues regions [13].

As we are dealing with curves, the possibility exists to explore additional data characteristics such as curvature. We performed an experiment with 1st order derivative functions, rather than the original functions to exploit potentially available higher order structure. Figure 4 (left) shows the cross correlation for 1st order derivative functions. The cross correlation values are similar to the ones shown in Figure 2. The average cross correlation value is slightly better than for the original data: 0.38. This may indicate that higher order structures may contain more predictive power.

In order to get insight in the effect of non-linear warping on prediction performance, we conducted an experiment in which we used convolutions of the stimulus $g(t)$ with different forms of a HRF function modeled by two gamma functions. Various HRF functions were obtained by varing the delay of response (relative to onset), delay of undershoot (relative to onset), dispersion of response, dispersion of undershoot, ratio of response to undershoot. To determine $g_s(t)$, we convolved $g(t)$ with 16 different HRF functions, and selected the convolved one with highest cross correlation with $f_s(t)$ to be $g_s(t)$. Hence, we parametrically modeled the HRF and learned its parameters from the data.

Figure 4 (right) shows the results of the experiments with convolution of stimuli data with HRF models learned from the data. As can be seen, the cross correlation values are much lower compared to the values in Figure 2 (left). The average cross correlation value is 0.31. Hence, non-linear warping of stimulus onto voxel time course significantly enhances the predictive power of our model. This suggests that non-linear warping is a potential alternative for determining the best possible HRF estimate to overcome potential negative consequences of assuming HRF consistency across subjects or brain regions [14].

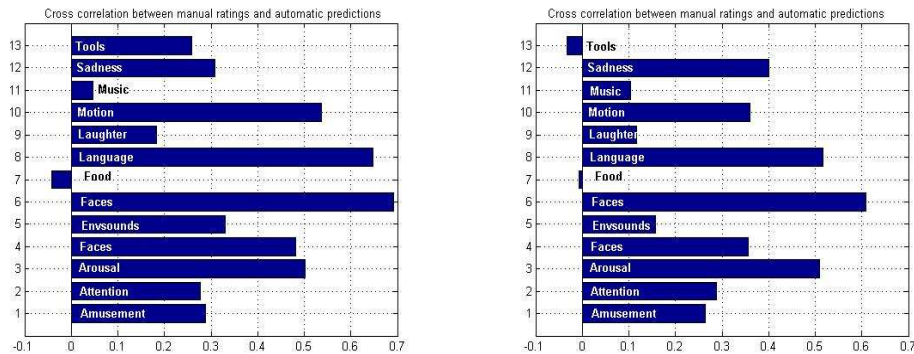

Figure 4: Left: normalized cross correlation values from cross-validation for 13 core movie features, using 1st order derivative data. Right: cross correlation values from cross-validation for 13 core movie features, using HRF convoluted rather than warped stimuli data.

## 4 Conclusion

Functional data analysis provides the possibility to fully exploit structure in inherently continuous data such as fMRI. The advantage of functional data analysis for principal component analysis of fMRI data was recently demonstrated in [10]. Here, we proposed a functional linear model that treats fMRI and stimuli as stochastic functional measurements. Cast into an incremental pattern searching framework, the method provides the ability to identify important covariance structure

of spatially distributed brain responses and stimuli, i.e. it directly couples activation across brain regions rather than first localizing and then integrating function. The method is suited for unbiased probing of functional characteristics of brain areas as well as for exposing meaningful relations between complex stimuli and distributed brain responses. This finding is supported by the good prediction performance of our method in the 2006 PBAIC international competition for brain activity interpretation. We are currently extending the method with new objective functions, dimension reduction techniques and multi-target search techniques to cope with multiple (interacting) stimuli. Also, in this work we made use of spatial clusters at a single hierarchical level. Preliminary results with hierarchical clustering to arrive at "supervoxels" at different spatial resolutions, seem to further improve prediction power.

## References

[1] J. Haynes and G. Rees. Decoding mental states from brain activity in humans. *Nature Neuroscience*, 7(8):523–534, 2006.

[2] J. Haynes and G. Rees. Predicting the orientation of invisible stimuli from activity in human primary visual cortex. *Nature Neuroscience*, 7(5):686–691, 2005.

[3] Y. Kamitani and F. Tong. Decoding the visual and subjective contents of the human brain. *Nature Neuroscience*, 8(5):679–685, 2005.

[4] S.M. Polyn, V.S. Natu, J.D. Cohen, and K.A. Norman. Category-specific cortical activity precedes retrieval during memory search. *Science*, 310(5756):1963–1966, 2005.

[5] T.M. Mitchell, R. Hutchinson, R.S. Niculescu, F. Pereira, X. Wang, M. Just, and S. Newman. Learning to decode cognitive states from brain images. *Machine Learning*, 57(1-2), 2004.

[6] W. Schneider, A. Bartels, E. Formisano, J. Haxby, R. Goebel, T. Mitchell, T. Nichols, and G. Siegle. Competition: Inferring experience based cognition from fmri. In *Proceedings Organization of Human Brain Mapping Florence Italy June 15*, 2006.

[7] Editorial. What's on your mind. *Nature Neuroscience*, 6(8):981, 2006.

[8] K.J. Worsley, J.B. Poline, K.J. Friston, and A.C. Evans. Characterizing the response of pet and fmri data using multivariate linear models. *Neuroimage*, 6, 1997.

[9] J. Ramsay and B. Silverman. *Functional Data Analysis*. Springer-Verlag, 1997.

[10] R. Viviani, G. Grohn, and M. Spitzer. Functional principal component analysis of fmri data. *Human Brain Mapping*, 24:109–129, 2005.

[11] D.B. Rowe and R.G. Hoffmann. Multivariate statistical analysis in fmri. *IEEE Engineering in Medicine and Biology*, 25:60–64, 2006.

[12] Shumeet Baluja. Population-based incremental learning: A method for integrating genetic search based function optimization and competitive learning. Technical Report CMU-CS-94-163, Computer Science Department, Carnegie Mellon University, Pittsburgh, PA, 1994.

[13] M.A. Gernsbacher and M.P. Kaschak. Neuroimaging studies of language production and comprehension. *Annual Review of Psychology*, 54:91–114, 2003.

[14] D.A. Handwerker, J.M. Ollinger, and M. D'Esposito. Variation of bold hemodynamic response function across subjects and brain regions and their effects on statistical analysis. *NeuroImage*, 8(21):1639–1651, 2004.
